# Discriminative Clustering by Regularized Information Maximization

**Ryan Gomes**
gomes@vision.caltech.edu

**Andreas Krause**
krausea@caltech.edu

**Pietro Perona**
perona@vision.caltech.edu
California Institute of Technology
Pasadena, CA 91106

## Abstract

Is there a principled way to learn a probabilistic discriminative classifier from an unlabeled data set? We present a framework that simultaneously clusters the data and trains a discriminative classifier. We call it *Regularized Information Maximization* (RIM). RIM optimizes an intuitive information-theoretic objective function which balances class separation, class balance and classifier complexity. The approach can flexibly incorporate different likelihood functions, express prior assumptions about the relative size of different classes and incorporate partial labels for semi-supervised learning. In particular, we instantiate the framework to unsupervised, multi-class kernelized logistic regression. Our empirical evaluation indicates that RIM outperforms existing methods on several real data sets, and demonstrates that RIM is an effective model selection method.

## 1   Introduction

Clustering algorithms group data items into categories without requiring human supervision or definition of categories. They are often the first tool used when exploring new data. A great number of clustering principles have been proposed, most of which can be described as either *generative* or *discriminative* in nature. Generative clustering algorithms provide constructive definitions of categories in terms of their geometric properties in a feature space or as statistical processes for generating data. Examples include k-means and Gaussian mixture model clustering. In order for generative clustering to be practical, restrictive assumptions must be made about the underlying category definitions.

Rather than modeling categories explicitly, discriminative clustering techniques represent the boundaries or distinctions between categories. Fewer assumptions about the nature of categories are made, making these methods powerful and flexible in real world applications. Spectral graph partitioning [1] and maximum margin clustering [2] are example discriminative clustering methods. A disadvantage of existing discriminative approaches is that they lack a probabilistic foundation, making them potentially unsuitable in applications that require reasoning under uncertainty or in data exploration.

We propose a principled probabilistic approach to discriminative clustering, by formalizing the problem as unsupervised learning of a conditional probabilistic model. We generalize the work of Grandvalet and Bengio [3] and Bridle et al. [4] in order to learn probabilistic classifiers that are appropriate for multi-class discriminative clustering, as explained in Section 2. We identify two fundamental, competing quantities, class balance and class separation, and develop an information theoretic objective function which trades off these quantities. Our approach corresponds to maximizing mutual information between the empirical distribution on the inputs and the induced

label distribution, regularized by a complexity penalty. Thus, we call our approach Regularized Information Maximization (RIM).

In summary, our contribution is RIM, a probabilistic framework for discriminative clustering with a number of attractive properties. Thanks to its probabilistic formulation, RIM is flexible: it is compatible with diverse likelihood functions and allows specification of prior assumptions about expected class proportions. We show how our approach leads to an efficient, scalable optimization procedure that also provides a means of automatic model selection (determination of the number of clusters). RIM is easily extended to semi-supervised classification. Finally, we show that RIM performs better than competing approaches on several real-world data sets.

## 2 Regularized Information Maximization

Suppose we are given an unlabeled dataset of $N$ feature vectors (datapoints) $\mathbf{X} = (\mathbf{x}_1, \cdots, \mathbf{x}_N)$, where $\mathbf{x}_i = (x_{i1}, \ldots, x_{iD})^T \in \mathbb{R}^D$ are $D$-dimensional vectors with components $x_{id}$. Our goal is to learn a conditional model $p(y|\mathbf{x}, \mathbf{W})$ with parameters $\mathbf{W}$ which predicts a distribution over label values $y \in \{1, \ldots, K\}$ given an input vector $\mathbf{x}$.

Our approach is to construct a functional $F(p(y|\mathbf{x}, \mathbf{W}); \mathbf{X}, \lambda)$ which evaluates the suitability of $p(y|\mathbf{x}, \mathbf{W})$ as a discriminative clustering model. We then use standard discriminative classifiers such as logistic regression for $p(y|\mathbf{x}, \mathbf{W})$, and maximize the resulting function $F(\mathbf{W}; \mathbf{X}, \lambda)$ over the parameters $\mathbf{W}$. $\lambda$ is an additional tuning parameter that is fixed during optimization.

We are guided by three principles when constructing $F(p(y|\mathbf{x}, \mathbf{W}); \mathbf{X}, \lambda)$. The first is that the discriminative model's decision boundaries should not be located in regions of the input space that are densely populated with datapoints. This is often termed the *cluster assumption* [5], and also corresponds to the idea that datapoints should be classified with large margin. Grandvalet & Bengio [3] show that a conditional entropy term $-\frac{1}{N} \sum_i H\{p(y|\mathbf{x}_i, \mathbf{W})\}$ very effectively captures the cluster assumption when training probabilistic classifiers with partial labels. However, in the case of fully unsupervised learning this term alone is not enough to ensure sensible solutions, because conditional entropy may be reduced by simply removing decision boundaries and unlabeled categories tend to be removed. We illustrate this in Figure 1 (left) with an example using the multilogit regression classifier as the conditional model $p(y|\mathbf{x}, \mathbf{W})$, which we will develop in Section 3.

In order to avoid degenerate solutions, we incorporate the notion of class balance: we prefer configurations in which category labels are assigned evenly across the dataset. We define the empirical label distribution

$$\hat{p}(y; \mathbf{W}) = \int \hat{p}(\mathbf{x}) p(y|\mathbf{x}, \mathbf{W}) d\mathbf{x} = \frac{1}{N} \sum_i p(y|\mathbf{x}_i, \mathbf{W}),$$

which is an estimate of the marginal distribution of $y$. A natural way to encode our preference towards class balance is to use the entropy $H\{\hat{p}(y; \mathbf{W})\}$, because it is maximized when the labels are uniformly distributed. Combining the two terms, we arrive at

$$I_\mathbf{W}\{y; \mathbf{x}\} = H\{\hat{p}(y; \mathbf{W})\} - \frac{1}{N} \sum_i H\{p(y|\mathbf{x}_i, \mathbf{W})\} \tag{1}$$

which is the empirical estimate of the mutual information between $\mathbf{x}$ and $y$ under the conditional model $p(y|\mathbf{x}, \mathbf{W})$.

Bridle et al. [4] were the first to propose maximizing $I_\mathbf{W}\{y; \mathbf{x}\}$ in order to learn probabilistic classifiers without supervision. However, they note that $I_\mathbf{W}\{y; \mathbf{x}\}$ may be trivially maximized by a conditional model that classifies each data point $\mathbf{x}_i$ into its own category $y_i$, and that classifiers trained with this objective tend to fragment the data into a large number of categories, see Figure 1 (center). We therefore introduce a regularizing term $R(\mathbf{W}; \lambda)$ whose form will depend on the specific choice of $p(y|\mathbf{x}, \mathbf{W})$. This term penalizes conditional models with complex decision boundaries in order to yield sensible clustering solutions. Our objective function is

$$F(\mathbf{W}; \mathbf{X}, \lambda) = I_\mathbf{W}\{y; \mathbf{x}\} - R(\mathbf{W}; \lambda) \tag{2}$$

and we therefore refer to our approach as *Regularized Information Maximization* (RIM), see Figure 1 (right). While we motivated this objective with notions of class balance and seperation, our approach may be interpreted as learning a conditional distribution for $y$ that preserves information from the data set, subject to a complexity penalty.

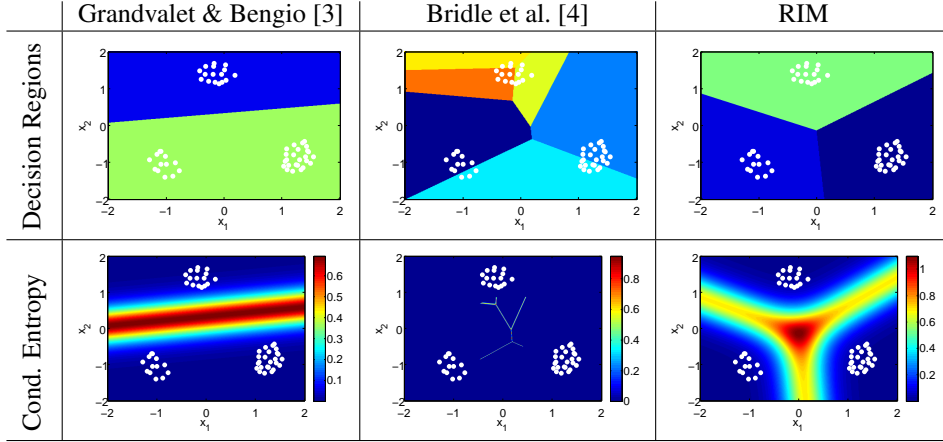

Figure 1: Example unsupervised multilogit regression solutions on a simple dataset with three clusters. The top and bottom rows show the category label $\arg\max_y p(y|\mathbf{x}, \mathbf{W})$ and conditional entropy $H\{p(y|\mathbf{x}, \mathbf{W})\}$ at each point $\mathbf{x}$, respectively. We find that both class balance and regularization terms are necessary to learn unsupervised classifiers suitable for multi-class clustering.

## 3  Example application: Unsupervised Multilogit Regression

The RIM framework is flexible in the choice of $p(y \mid \mathbf{x}; \mathbf{W})$ and $R(\mathbf{W}; \lambda)$. As an example instantiation, we here choose multiclass logistic regression as the conditional model. Specifically, if $K$ is the maximum number of classes, we choose

$$p(y = k|\mathbf{x}, \mathbf{W}) \propto \exp(\mathbf{w}_k^T \mathbf{x} + b_k) \quad \text{and} \quad R(\mathbf{W}; \lambda) = \lambda \sum_k \mathbf{w}_k^T \mathbf{w}_k, \tag{3}$$

where the set of parameters $\mathbf{W} = \{\mathbf{w}_1, \ldots, \mathbf{w}_K; b_1, \ldots, b_K\}$ consists of weight vectors $\mathbf{w}_k$ and bias values $b_k$ for each class $k$. Each weight vector $\mathbf{w}_k \in \mathbb{R}^D$ is $D$-dimensional with components $w_{kd}$. The regularizer is the squared $L_2$ norm of the weight vectors, and may be interpreted as an isotropic normal distribution prior on the weights $\mathbf{W}$. The bias terms are not penalized.

In order to optimize Eq. 2 specialized with Eqs. 3, we require the gradients of the objective function. For clarity, we define $p_{ki} \equiv p(y = k|\mathbf{x}_i, \mathbf{W})$, and $\hat{p}_k \equiv \hat{p}(y = k; \mathbf{W})$. The partial derivatives are

$$\frac{\partial F}{\partial w_{kd}} = \frac{1}{N} \sum_{ic} \frac{\partial p_{ci}}{\partial w_{kd}} \log \frac{p_{ci}}{\hat{p}_c} - 2\lambda w_{kd} \quad \text{and} \quad \frac{\partial F}{\partial b_k} = \frac{1}{N} \sum_{ic} \frac{\partial p_{ci}}{\partial b_k} \log \frac{p_{ci}}{\hat{p}_c}. \tag{4}$$

Naive computation of the gradient requires $O(NK^2D)$, since there are $K(D + 1)$ parameters and each derivative requires a sum over $NK$ terms. However, the form of the conditional probability derivatives for multi-logit regression are:

$$\frac{\partial p_{ci}}{\partial w_{kd}} = (\delta_{kc} - p_{ci})p_{ki}x_{id} \quad \text{and} \quad \frac{\partial p_{ci}}{\partial b_k} = (\delta_{kc} - p_{ci})p_{ki},$$

where $\delta_{kc}$ is equal to one when indices $k$ and $c$ are equal, and zero otherwise. When these expressions are substituted into Eq. 4, we find the following expressions:

$$\frac{\partial F}{\partial w_{kd}} = \frac{1}{N} \sum_i x_{id} p_{ki} \left( \log \frac{p_{ki}}{\hat{p}_k} - \sum_c p_{ci} \log \frac{p_{ci}}{\hat{p}_c} \right) - 2\lambda w_{kd} \tag{5}$$

$$\frac{\partial F}{\partial b_k} = \frac{1}{N} \sum_i p_{ki} \left( \log \frac{p_{ki}}{\hat{p}_k} - \sum_c p_{ci} \log \frac{p_{ci}}{\hat{p}_c} \right)$$

Computing the gradient requires only $O(NKD)$ operations since the terms $\sum_c p_{ci} \log \frac{p_{ci}}{\hat{p}_c}$ may be computed once and reused in each partial derivative expression.

The above gradients are used in the L-BFGS [6] quasi-Newton optimization algorithm[1]. We find empirically that the optimization usually converges within a few hundred iterations. When specialized

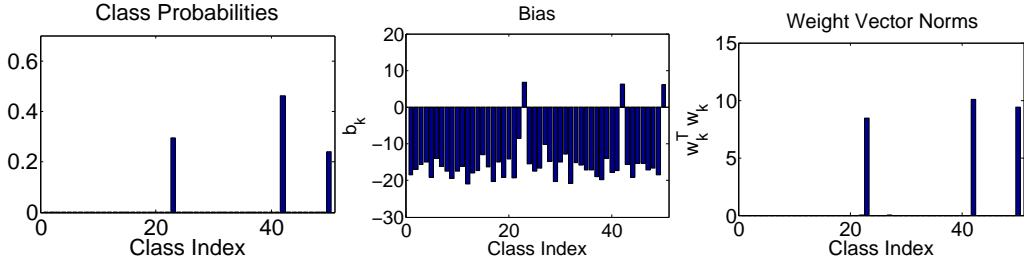

Figure 2: Demonstration of model selection on the toy problem from Figure 1. The algorithm is initialized with 50 category weight vectors $\mathbf{w}_k$. Upon convergence, only three of the categories are populated with data examples. The negative bias terms of the unpopulated categories drive the unpopulated class probabilities $\hat{p}_k$ towards zero. The corresponding weight vectors $\mathbf{w}_k$ have norms near zero.

to multilogit regression, the objective function $F(\mathbf{W}; \mathbf{x}, \lambda)$ is non-concave. Therefore the algorithm can only be guaranteed to halt at locally optimal stationary points of $F$. In Section 3.1, we explain how we can obtain an initialization that is robust against local optima.

## 3.1 Model Selection

Setting the derivatives (Eq. 5) equal to zero yields the following condition at stationary points of $F$:

$$\mathbf{w}_k = \sum_i \alpha'_{ki} \mathbf{x}_i \qquad (6)$$

where we have defined

$$\alpha'_{ki} \equiv \frac{1}{2\lambda N} p_{ki} \left( \log \frac{p_{ki}}{\hat{p}_k} - \sum_c p_{ci} \log \frac{p_{ci}}{\hat{p}_c} \right). \qquad (7)$$

The $L_2$ regularizing function $R(\mathbf{W}; \lambda)$ in Eq. 3 is additively composed of penalty terms associated with each category: $\mathbf{w}_k^T \mathbf{w}_k = \sum_{ij} \alpha'_{ki} \alpha'_{kj} \mathbf{x}_i^T \mathbf{x}_j$. It is instructive to observe the limiting behavior of the penalty term $\mathbf{w}_k^T \mathbf{w}_k$ when datapoints are not assigned to category $k$; that is, when $\hat{p}_k = \frac{1}{N} \sum_i p_{ki} \to 0$. This implies that $p_{ki} \to 0$ for all $i$, and therefore $\alpha'_{ki} \to 0$ for all $i$. Finally, $\mathbf{w}_k^T \mathbf{w}_k = \sum_{ij} \alpha'_{ki} \alpha'_{kj} \mathbf{x}_i^T \mathbf{x}_j \to 0$. This means that the regularizing function does not penalize unpopulated categories.

We find empirically that when we initialize with a large number of category weights $\mathbf{w}_k$, many decay away depending on the value of $\lambda$. Typically as $\lambda$ increases, fewer categories are discovered. This may be viewed as model selection (automatic determination of the number of categories) since the regularizing function and parameter $\lambda$ may be interpreted as a form of prior on the weight parameters. The bias terms $b_k$ are unpenalized and are adjusted during optimization to drive the class probablities $\hat{p}_k$ arbitrarily close to zero for unpopulated classes. This is illustrated in Figure 2.

This behavior suggests an effective initialization procedure for our algorithm. We first oversegment the data into a large number of clusters (using k-means or other suitable algorithm) and train a *supervised* multi-logit classifier using these cluster labels. (This initial classifier may be trained with a small number of L-BFGS iterations since it only serves as a starting point.) We then use this classifier as the starting point for our RIM algorithm and optimize with different values of $\lambda$ in order to obtain solutions with different numbers of clusters.

## 4 Example Application: Unsupervised Kernel Multilogit Regression

The stationary conditions have another interesting consequence. Equation 6 indicates that at stationary points, the weights are located in the span of the input datapoints. We use this insight as justification to define explicit coefficients $\alpha_{ki}$ and enforce the constraint $\mathbf{w}_k = \sum_i \alpha_{ki} \mathbf{x}_i$ during optimization. Substituting this equation into the multilogit regression conditional likelihood allows replacement of all inner products $\mathbf{w}_k^T \mathbf{x}$ with $\sum_i \alpha_{ki} K(\mathbf{x}_i, \mathbf{x})$, where $K$ is a positive definite kernel function that evaluates the inner product $\mathbf{x}_i^T \mathbf{x}$. The conditional model now has the form

$$p(y = k | \mathbf{x}, \alpha, \mathbf{b}) \propto \exp\left( \sum_i \alpha_{ki} K(\mathbf{x}_i, \mathbf{x}) + b_k \right).$$

Substituting the constraint into the regularizing function $\sum_k \mathbf{w}_k^T \mathbf{w}_k$ yields a natural replacement of $\mathbf{w}_k^T \mathbf{w}_k$ by the Reproducing Hilbert Space (RKHS) norm of the function $\sum_i \alpha_{ki} K(\mathbf{x}_i, \cdot)$:

$$R(\alpha) = \sum_k \sum_{ij} \alpha_{ki} \alpha_{kj} K(\mathbf{x}_i, \mathbf{x}_j). \tag{8}$$

We use the L-BFGS algorithm to optimize the kernelized algorithm over the coefficients $\alpha_{ki}$ and biases $b_k$. The partial derivatives for the kernel coefficients are

$$\frac{\partial F}{\partial \alpha_{kj}} = \frac{1}{N} \sum_i K(\mathbf{x}_j, \mathbf{x}_i) p_{ki} \left( \log \frac{p_{ki}}{\hat{p}_k} - \sum_c p_{ci} \log \frac{p_{ci}}{\hat{p}_c} \right) - 2\lambda \sum_i \alpha_{ki} K(\mathbf{x}_j, \mathbf{x}_i)$$

and the derivatives for the biases are unchanged. The gradient of the kernelized algorithm requires $O(KN^2)$ to compute. Kernelized unsupervised multilogit regression exhibits the same model selection behavior as the linear algorithm.

# 5 Extensions

We now discuss how RIM can be extended to semi-supervised classification, and to encode prior assumptions about class proportions.

## 5.1 Semi-supervised Classification

In semi-supervised classification, we assume that there are unlabeled examples $\mathbf{X}^U = \{\mathbf{x}_1^U, \cdots, \mathbf{x}_N^U\}$ as well as labeled examples $\mathbf{X}^L = \{\mathbf{x}_1^L, \cdots, \mathbf{x}_M^L\}$ with labels $Y = \{y_1, \cdots, y_M\}$.

We again use mutual information $I_{\mathbf{W}}\{y; \mathbf{x}\}$ (Eq. 1) to define the relationship between unlabeled points and the model parameters, but we incorporate an additional parameter $\tau$ which will define the tradeoff between labeled and unlabeled examples. The conditional likelihood is incorporated for labeled examples to yield the semi-supervised objective:

$$S(\mathbf{W}; \tau, \lambda) = \tau I_{\mathbf{W}}\{y; \mathbf{x}\} - R(\mathbf{W}; \lambda) + \sum_i \log p(y_i | \mathbf{x}_i^L, \mathbf{W})$$

The gradient is computed and again used in the L-BFGS algorithm in order to optimize this combined objective. Our approach is related to the objective in [3], which does not contain the class balance term $H(\hat{p}(y; \mathbf{W}))$.

## 5.2 Encoding Prior Beliefs about the Label Distribution

So far, we have motivated our choice for the objective function $F$ through the notion of class balance. However, in many classification tasks, different classes have different number of members. In the following, we show how RIM allows flexible expression of prior assumptions about non-uniform class label proportions.

First, note that the following basic identity holds

$$H\{\hat{p}(y; \mathbf{W})\} = \log(K) - KL\{\hat{p}(y; \mathbf{W}) \| U\} \tag{9}$$

where $U$ is the uniform distribution over the set of labels $\{1, \cdots, K\}$. Substituting the identity, then dropping the constant $\log(K)$ yields another interpretation of the objective

$$F(\mathbf{W}; \mathbf{X}, \lambda) = -\frac{1}{N} \sum_i H\{p(y | \mathbf{x}_i, \mathbf{W})\} - KL\{\hat{p}(y; \mathbf{W}) \| U\} - R(\mathbf{W}; \lambda). \tag{10}$$

The term $-KL\{\hat{p}(y; \mathbf{W}) \| U\}$ is maximized when the average label distribution is uniform. We can capture prior beliefs about the average label distribution by substituting a reference distribution $D(y; \gamma)$ in place of $U$ ($\gamma$ is a parameter that may be fixed or optimized during learning). [7] also use relative entropy as a means of enforcing prior beliefs, although not with respect to class distributions in multi-class classification problems.

This construction may be used in a clustering task in which we believe that the cluster sizes obey a power law distribution as, for example, considered by [8] who use the Pitman-Yor process for nonparametric language modeling. Simple manipulation yields the following objective:

$$F(\mathbf{W}; \mathbf{X}, \lambda, \gamma) = I_{\mathbf{W}}\{\mathbf{x}; y\} - H\{\hat{p}(y; \mathbf{W}) \| D(y; \gamma)\} - R(\mathbf{W}; \lambda)$$

where $H\{\hat{p}(y; \mathbf{W}) \| D(y; \gamma)\}$ is the cross entropy $-\sum_k \hat{p}(y = k; \mathbf{W}) \log D(y = k; \gamma)$. We therefore find that label distribution priors may be incorporated using an additional cross entropy regularization term.

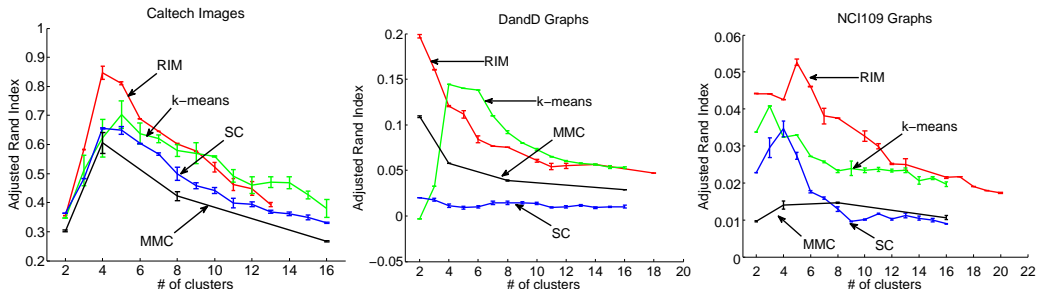

Figure 3: Unsupervised Clustering: Adjusted Rand Index (relative to ground truth) versus number of clusters.

## 6 Experiments

We empirically evaluate our RIM approach on several real data sets, in both fully unsupervised and semisupervised configurations.

### 6.1 Unsupervised Learning

Kernelized RIM is initialized according to the procedure outlined in Section 3.1, and run until L-BFGS converges. Unlabeled examples are then clustered according to $\arg\max_k p(y = k|\mathbf{x}, \mathbf{W})$. We compare RIM against the spectral clustering (SC) algorithm of [1], the fast maximum margin clustering (MMC) algorithm of [9], and kernelized k-means [10]. MMC is a binary clustering algorithm. We use the recursive scheme outlined by [9] to extend the approach to multiple categories. The MMC algorithm requires an initial clustering estimate for initialization, and we use SC to provide this.

We evaluate unsupervised clustering performance in terms of how well the discovered clusters reflect known ground truth labels of the dataset. We report the Adjusted Rand Index (ARI) [11] between an inferred clustering and the ground truth categories. ARI has a maximum value of 1 when two clusterings are identical. We evaluated a number of other measures for comparing clusterings to ground truth including mutual information, normalized mutual information [12], and cluster impurity [13]. We found that the relative rankings of the algorithms were the same as indicated by ARI.

We evaluate the performance of each algorithm while varying the number of clusters that are discovered, and we plot ARI for each setting. For SC and k-means the number of clusters is given as an input parameter. MMC is evaluated at $\{2, 4, 8, \cdots\}$ clusters (powers of two, due to the recursive scheme.) For RIM, we sweep the regularization parameter $\lambda$ and allow the algorithm to discover the final number of clusters.

**Image Clustering.** We test the algorithms on an image clustering task with 350 images from four Caltech-256 [14] categories (Faces-Easy, Motorbikes, Airplanes, T-Shirt) for a total of $N = 1400$ images. We use the Spatial Pyramid Match kernel [15] computed between every pair of images. We sweep RIM's $\lambda$ parameter across $\left[\frac{0.125}{N}, \frac{4}{N}\right]$. The results are summarized in figure 3. Overall, the clusterings that best match ground truth are given by RIM when it discovers four clusters. We find that RIM outperforms both SC and MMC at all settings. RIM outperforms kernelized k-means when discovering between 4 and 8 clusters. Their performances are comparable for other numbers of clusters. Figure 4 shows example images taken from clusters discovered by RIM. Our RIM implementation takes approximately 110 seconds per run on the Caltech Images datset on a quad core Intel Xeon server. SC requires 38 seconds per run, while MMC requires 44-51 seconds per run depending on the number of clusters specified.

**Molecular Graph Clustering.** We further test RIM's unsupervised learning performance on two molecular graph datasets. D&D [16] contains $N = 1178$ protein structure graphs with binary ground truth labels indicating whether or not they function as enzymes. NCI109 [17] is composed of $N = 4127$ compounds labeled according to whether or not they are active in an anti-cancer screening. We use the subtree kernel developed by [18] with subtree height of 1. For D&D, we sweep RIM's lambda parameter through the range $\left[\frac{0.001}{N}, \frac{0.05}{N}\right]$ and for NCI we sweep through the interval $\left[\frac{0.001}{N}, \frac{1}{N}\right]$. Results are summarized in Figures 3 (center and right). We find that of all methods, RIM produces the clusterings that are nearest to ground truth (when discovering 2 clusters

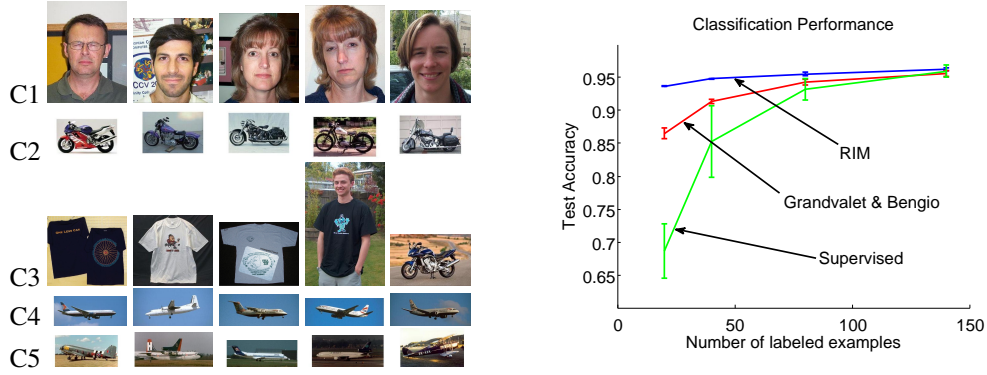

C1
C2
C3
C4
C5

Classification Performance

Figure 4: Left: Randomly chosen example images from clusters discovered by unsupervised RIM on Caltech Image. Right: Semi-supervised learning on Caltech Images.

| Average Waveform | Most Uncertain Waveform |
| --- | --- |

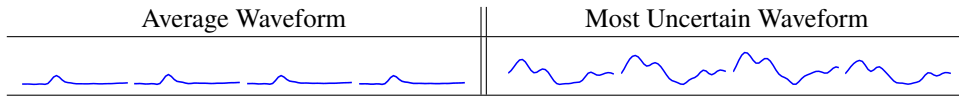

Figure 5: Left, Tetrode dataset average waveform. Right, the waveform with the most uncertain cluster membership according to the classifier learned by RIM.

for D&D and 5 clusters for NCI109). RIM outperforms both SC and MMC at all settings. RIM has the advantage over k-means when discovering a small number of clusters and is comparable at other settings. On NCI109, RIM required approximately 10 minutes per run. SC required approximately 13 minutes, while MMC required on average 18 minutes per run.

**Neural Tetrode Recordings.** We demonstrate RIM on a large scale data set of $319,209$ neural activity waveforms recorded from four co-located electrodes implanted in the hippocampus of a behaving rat. The waveforms are composed of 38 samples from each of the four electrodes and are the output of a neural spike detector which aligns signal peaks to the 13-th sample, see the average waveform in Figure 5 (left). We concatenate the samples into a single 152-dimensional vector and preprocess by subtracting the mean waveform and divide each vector component by its variance. We use the linear RIM algorithm given in Section 3, initialized with 100 categories. We set $\lambda$ to $\frac{4}{N}$ and RIM discovers 33 clusters and finishes in 12 minutes. There is no ground truth available for this dataset, but we use it to demonstrate RIM's efficacy as a data exploration tool. Figure 6 shows two clusters discovered by RIM. The top row consists of cluster member waveforms superimposed on each other, with the cluster's mean waveform plotted in red. We find that the clustered waveforms have substantial similarity to each other. Taken as a whole, the clusters give an idea of the typical waveform patterns. The bottom row shows the learned classifier's discriminative weights $\mathbf{w}_k$ for each category, which can be used to gain a sense for how the cluster's members differ from the dataset mean waveform. We can use the probabilistic classifier learned by RIM to discover atypical waveforms by ranking them according to their conditional entropy $H\{p(y|\mathbf{x}_i, \mathbf{W})\}$. Figure 5 (right) shows the waveform whose cluster membership is most uncertain.

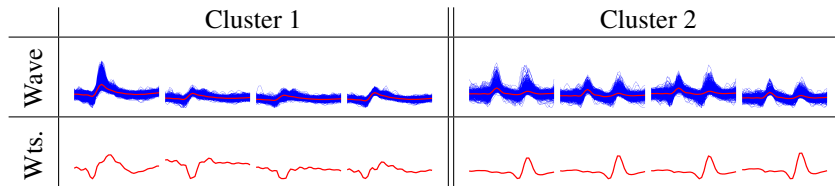

| | Cluster 1 | Cluster 2 |
| --- | --- | --- |
| Wave | | |
| Wts. | | |

Figure 6: Two clusters discovered by RIM on the Tetrode data set. Top row: Superimposed waveform members, with cluster mean in red. Bottom row: The discriminative category weights $\mathbf{w}_k$ associated with each cluster.

### 6.2 Semi-supervised Classification

We test our semi-supervised classification method described in Section 5.1 against [3] on the Caltech Images dataset. The methods were trained using both unlabeled and labeled examples, and classification performance is assessed on the unlabeled portion. As a baseline, a supervised classifier was trained on labeled subsets of the data and tested on the remainder. Parameters were selected via cross-validation on a subset of the labeled examples. The results are summarized in Figure 4. We find that both semi-supervised methods significantly improve classification performance relative to the supervised baseline when the number of labeled examples is small. Additionally, we find that RIM outperforms Grandvalet & Bengio. This suggests that incorporating prior knowledge about class size distributions (in this case, we use a uniform prior) may be useful in semi-supervised learning.

## 7 Related Work

Our work has connections to existing work in both unsupervised learning and semi-supervised classification.

**Unsupervised Learning.** The information bottleneck method [19] learns a conditional model $p(y|x)$ where the labels $y$ form a lossy representation of the input space $x$, while preserving information about a third "relevance" variable $z$. The method maximizes $I(y;z) - \lambda I(x;y)$, whereas we maximize the information between $y$ and $x$ while constraining complexity with a parametric regularizer. The method of [20] aims to maximize a similarity measure computed between members within the same cluster while penalizing the mutual information between the cluster label $y$ and the input $x$. Again, mutual information is used to enforce a lossy representation of $y|x$. Song et al. [22] also view clustering as maximization of the dependence between the input variable and output label variable. They use the Hilbert-Schmidt Independence Criterion as a measure of dependence, whereas we use Mutual Information.

There is also an unsupervised variant of the Support Vector Machine, called max-margin clustering. Like our approach, the works of [2] and [21] use notions of class balance, seperation, and regularization to learn unsupervised discriminative classifiers. However, they are formulated in the max-margin framework rather than our probabilistic approach. Ours appears more amenable to incorporating prior beliefs about the class labels. Unsupervised SVMs are solutions to a convex relaxation of a non-convex problem, while we directly optimize our non-convex objective. The semidefinite programming methods required are much more expensive than our approach.

**Semi-supervised Classification.** Our semi-supervised objective is related to [3], as discussed in section 5.1. Another semi-supervised method [23] uses mutual information as a regularizing term to be minimized, in contrast to ours which attempts to maximize mutual information. The assumption underlying [23] is that any information between the label variable and unlabeled examples is an artifact of the classifier and should be removed. Our method encodes the opposite assumption: there may be variability (e.g. new class label values) not captured by the labeled data, since it is incomplete.

## 8 Conclusions

We considered the problem of learning a probabilistic discriminative classifier from an unlabeled data set. We presented Regularized Information Maximization (RIM), a probabilistic framework for tackling this challenge. Our approach consists of optimizing an intuitive information theoretic objective function that incorporates class separation, class balance and classifier complexity, which may be interpreted as maximizing the mutual information between the empirical input and implied label distributions. The approach is flexible, in that it allows consideration of different likelihood functions. It also naturally allows expression of prior assumptions about expected label proportions by means of a cross-entropy with respect to a reference distribution. Our framework allows natural incorporation of partial labels for semi-supervised learning. In particular, we instantiate the framework to unsupervised, multi-class kernelized logistic regression. Our empirical evaluation indicates that RIM outperforms existing methods on several real data sets, and demonstrates that RIM is an effective model selection method.

#### Acknowledgements

We thank Alex Smola for helpful comments and discussion, and Thanos Siapas for providing the neural tetrode data. This research was partially supported by NSF grant IIS-0953413, a gift from Microsoft Corporation, and ONR MURI Grant N00014-06-1-0734.

## Footnotes

[1] We used Mark Schmidt's implementation at `http://www.cs.ubc.ca/~schmidtm/Software/minFunc.html`.

# References

[1] A. Ng, M. I. Jordan, and Y. Weiss. On spectral clustering: Analysis and an algorithm. In *NIPS*, 2001.

[2] L. Xu and D. Schuurmans. Unsupervised and semi-supervised multi-class support vector machines. In *AAAI*, 2005.

[3] Y. Grandvalet and Y. Bengio. Semi-supervised learning by entropy minimization. In *NIPS*, 2004.

[4] John S. Bridle, Anthony J. R. Heading, and David J. C. MacKay. Unsupervised classifiers, mutual information and 'phantom targets'. In John E. Moody, Steve J. Hanson, and Richard P. Lippmann, editors, *Advances in Neural Information Processing Systems*, volume 4, pages 1096–1101. Morgan Kaufmann Publishers, Inc., 1992.

[5] Olivier Chapelle and Alexander Zien. Semi-supervised classification by low density separation, September 2004.

[6] D. C. Liu and J. Nocedal. On the limited memory BFGS method for large scale optimization. *Mathematical Programming*, 45:503–528, 1989.

[7] T. Jaakkola, M. Meila, and T. Jebara. Maximum entropy discrimination. In *NIPS*, 1999.

[8] Y. W. Teh. A hierarchical bayesian language model based on pitman-yor processes. In *ACL*, 2006.

[9] K. Zhang, I. W. Tsang, and J. T. Kwok. Maximum margin clustering made practical. In *ICML*, 2007.

[10] John Shawe-Taylor and Nello Cristianini. *Kernel Methods for Pattern Analysis*. Cambridge University Press, New York, NY, USA, 2004.

[11] Lawrence Hubert and Phipps Arabie. Comparing partitions. *Journal of Classification*, 2:193–218, 1985.

[12] Alexander Strehl and Joydeep Ghosh. Cluster ensembles — A knowledge reuse framework for combining multiple partitions. *Journal of Machine Learning Research*, 3:583–617, 2002.

[13] Y. Chen, J. Ze Wang, and R. Krovetz. CLUE: cluster-based retrieval of images by unsupervised learning. *IEEE Trans. Image Processing*, 14(8):1187–1201, 2005.

[14] G. Griffin, A. Holub, and P. Perona. Caltech-256 object category dataset. Technical Report 7694, California Institute of Technology, 2007.

[15] S. Lazebnik, C. Schmid, and J. Ponce. Beyond bags of features: Spatial pyramid matching for recognizing natural scene categories. In *CVPR*, 2006.

[16] P. D. Dobson and A. J. Doig. Distinguishing enzyme structures from non-enzymes without alignments. *J. Mol. Biol.*, 330:771–783, Jul 2003.

[17] Nikil Wale and George Karypis. Comparison of descriptor spaces for chemical compound retrieval and classification. In *ICDM*, pages 678–689, 2006.

[18] N. Shervashidze and K. M. Borgwardt. Fast subtree kernels on graphs. In *NIPS*, 2010.

[19] N. Tishby, F. C. Pereira, and W. Bialek. The information bottleneck method. *CoRR*, physics/0004057, 2000.

[20] N. Slonim, G. S. Atwal, G. Tkacik, and W. Bialek. Information-based clustering. *Proc Natl Acad Sci U S A*, 102(51):18297–18302, December 2005.

[21] Francis Bach and Zaïd Harchaoui. DIFFRAC: a discriminative and flexible framework for clustering. In John C. Platt, Daphne Koller, Yoram Singer, and Sam T. Roweis, editors, *NIPS*. MIT Press, 2007.

[22] Le Song, Alex Smola, Arthur Gretton, and Karsten M. Borgwardt. A dependence maximization view of clustering. In *ICML '07: Proceedings of the 24th international conference on Machine learning*, pages 815–822, New York, NY, USA, 2007. ACM.

[23] A. Corduneanu and T. Jaakkola. On information regularization. In *UAI*, 2003.

